# Network generalization for production: Learning and producing styled letterforms

**Igor Grebert**
541 Cutwater Ln.
Foster City, CA
94404

**David G. Stork**
Ricoh Calif. Research Cen.
2882 Sand Hill Rd.# 115
Menlo Park, CA 94025

**Ron Keesing**
Dept. Physiology
U. C. S. F.
San Francisco, CA
94143

**Steve Mims**
Electrical Engin.
Stanford U.
Stanford, CA
94305

## Abstract

We designed and trained a connectionist network to generate letterforms in a new font given just a *few* exemplars from that font. During learning, our network constructed a distributed internal representation of fonts as well as letters, despite the fact that each training instance exemplified *both* a font and a letter. It was necessary to have separate but interconnected hidden units for "letter" and "font" representations — several alternative architectures were not successful.

## 1. INTRODUCTION

Generalization from examples is central to the notion of cognition and intelligent behavior (Margolis, 1987). Much research centers on generalization in *recognition*, as in optical character recognition, speech recognition, and so forth. In all such cases, during the recognition event the information content of the representation is reduced; sometimes categorization is binary, representing just one bit of information. Thus the information reduction in answering "Is this symphony by Mozart?" is very large.

A different class of problems requires generalization for *production*, e.g., paint a portrait of Madonna in the style of Matisse. Here during the production event a very low informational input ("Madonna," and "Matisse") is used to create a very *high* informational output, including color, form, etc. on the canvas. Such problems are a type of analogy, and typically require the generalization system to abstract out invariants in both the instance being presented (e.g., Madonna) *and* the style (e.g., Matisse), and to integrate these representations in a meaningful way. This must be done despite the fact that the system is never taught explicitly the features that correspond to Matisse's style alone, nor to Madonna's face alone, and is never presented an example of both simultaneously.

To explore this class of analogy and production issues, we addressed the following problem, derived from Hofstadter (1985):

**Given just a few letters in a new font, draw the remaining letters.**

Connectionist networks have recently been applied to production problems such as music composition (Todd, 1989), but our task is somewhat different. Whereas in music composition, memory and context (in the form of recurrent connections in a network) are used for pattern generation (melody or harmony), we have no such temporal or other explicit context information during the production of letterforms.

## 2. DATA, NETWORK AND TRAINING

Figure 1 illustrates schematically our class of problems, and shows a subset of the data used to train our network. The general problem is to draw all the remaining letterforms in a given font, such that those forms are recognizable as letters in the style of that font.

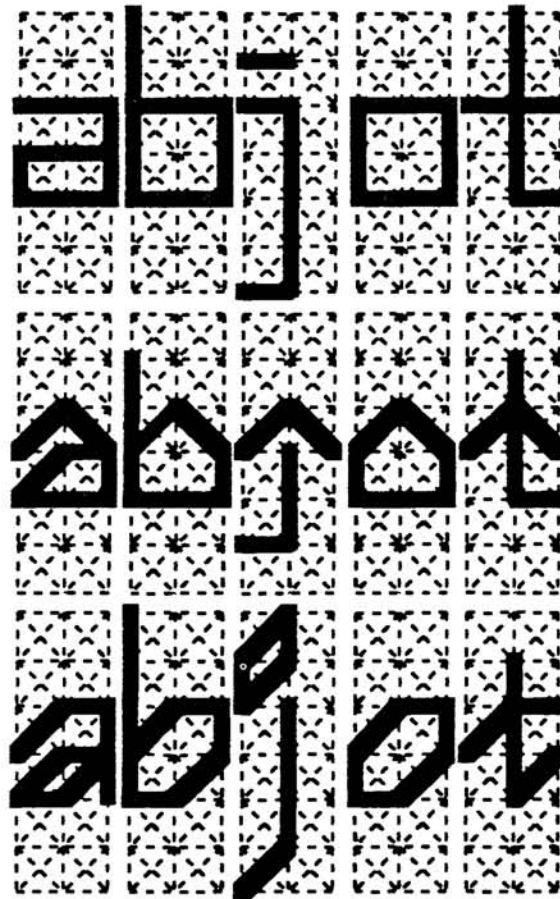

**Figure 1**: Several letters from three fonts (**Standard, House** and **Benzene right**) in Hofstadter's GridFont system. There are 56 fundamental horizontal, vertical and diagonal strokes, or "pixels," in the grid.

Each letterform in Figure 1 has a recognizable letter identity and "style" (or font). Each letter (columns) shares some invariant features as does each font (rows), though it would be quite difficult to describe what is the "same" in each of the **a**'s for instance, or for all letters in **Benzene right** font.

We trained our network with 26 letters in each of five fonts (**Standard, House, Slant, Benzene right** and **Benzene left**), and just 14 letters in the "test" font (**Hunt four** font). The task of the network was to reconstruct the missing 12 letters in **Hunt four** font. We used a structured three-level network (Figure 2) in which letter identity was represented in a 1-of-26 code (e.g., 010000... → b), and the font identity was represented in a similar 1-of-6 code. The letterforms were represented as 56-element binary vectors, with **1**'s for each stroke comprising the character, and were provided to the output units by a teacher. (Note that this network is "upside-down" from the typical use of connectionist networks for *categorization*.) The two sections of the input layer were each fully connected to the hidden layer, but the hidden layer-to-output layer connections were restricted (Figures 3 and 4). Such restricted hidden-to-output projections helped to prevent the learning of spurious and meaningless correlations between strokes in widely separate grid regions. There are unidirectional one-to-many intra-hidden layer connections from the letter section to the font section within the hidden layer (Figure 3).

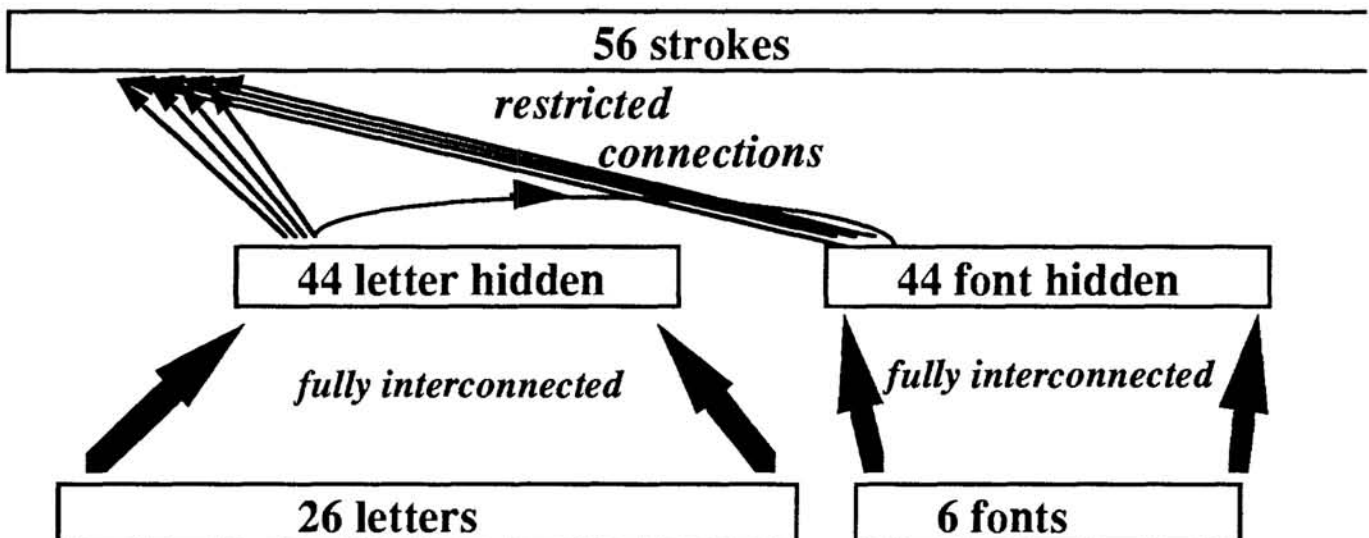

**Figure 2**: Network used for generalization in production. Note that the high-dimensional representation of strokes is at the *output* of the network, while the low-dimensional representation (a one-of-26 coding for letters and a one-of-six for fonts) is the *input*. The net has one-to-many connections from letter hidden units to font hidden units (cf. Figure 3)

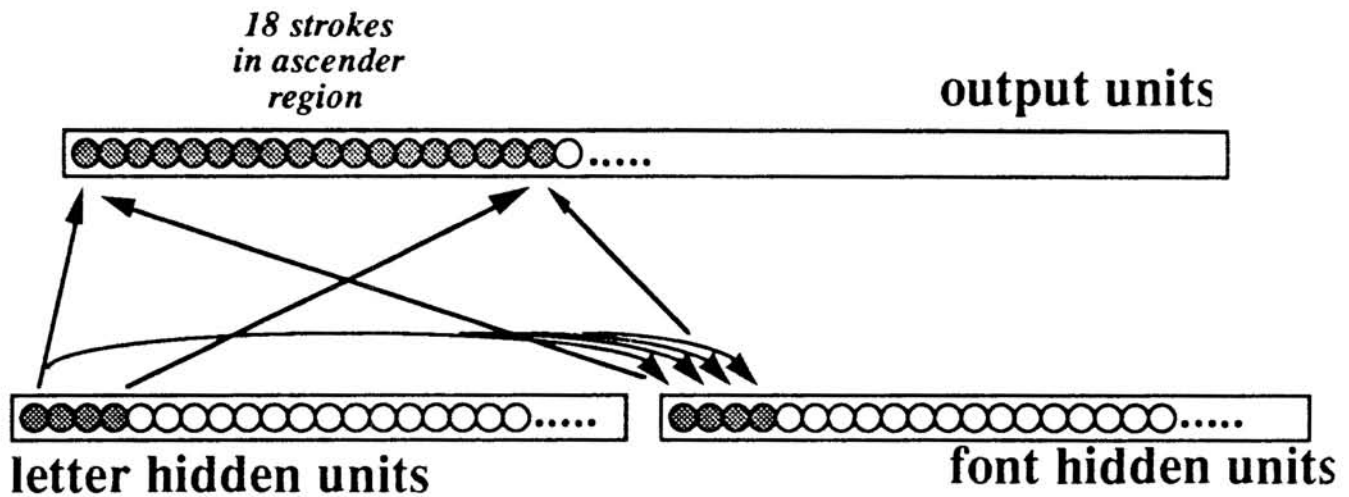

**18 strokes in ascender region**

**output units**

**letter hidden units**

**font hidden units**

**Figure 3**: Expanded view of the hidden and output layers of the network of Figure 2. Four letter hidden units and four font hidden units (dark) project fully to the eighteen stroke (output) units representing the ascender region of the GridFont grid; these hidden units project to no other output units. Each of the four letter hidden units also projects to all four of the corresponding font hidden units. This basic structure is repeated across the network (see text).

All connection weights, including intra-hidden layer weights, were adjusted using backpropagation (Rumelhart, Hinton and Williams, 1986), with a learning rate of $\eta = 0.005$ and momentum $\alpha = 0.9$. The training error stopped decreasing after roughly 10,000 training epochs, where each epoch consisted of one presentation of each of the 144 patterns (26 letters x 5 fonts + 14 letters) in random order.

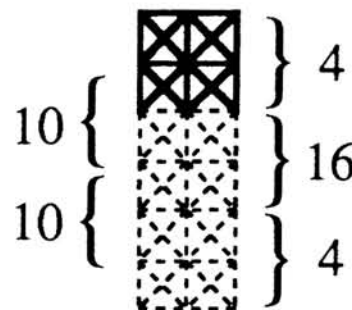

**Figure 4**: The number of hidden units projecting to each region of the output. Four font hidden units and four letter hidden units project to the 18 top strokes (ascender region) of the output layer, as indicated. Ten font hidden units and ten letter hidden units project to the next lower square region (20 strokes), etc. This restriction prevents the learning of meaningless correlations between particular strokes in the ascender and descender regions (for instance). Such spurious correlations disrupt learning and generalization only with a small training set such as ours.

## 3. RESULTS AND CONCLUSIONS

In order to produce any letterform, we presented as input to the trained network a (very sparse) 1-of-26 and 1-of-6 signal representing the target letter and font; the letterforms emerged at the output layer. Our network reproduced nearly perfectly all the patterns in the training set.

Figure 5 shows *untrained* letterforms generated by the network. Note that despite irregularities, all the letters except **z** can be easily recognized by humans. Moreover, the letterforms typically share the common style of **Hunt four** font — **b, c, g,** and **p** have the diamond-shaped "loop" of **o, q,** and other letters in the font; the **g** and **y** generated have the same right descender, similar to that in several letters of the original font, and so on; the **I** exactly matches the form designed by Hofstadter. Incidentally, we found that some of the letterforms produced by the network could be considered superior to those designed by Hofstadter. For instance, the generated **w** had the characteristic **Hunt four** diamond shape while the **w** designed by Hostadter did not. We must stress, though, that there is no "right" answer here; the letterforms provided by Hofstadter are merely one possible solution. Just as there is no single "correct" portrait of Madonna in the style of Matisse, so our system must be judged successful if the letterforms produced are both legible and have the style implied by the other letterforms in the test font.

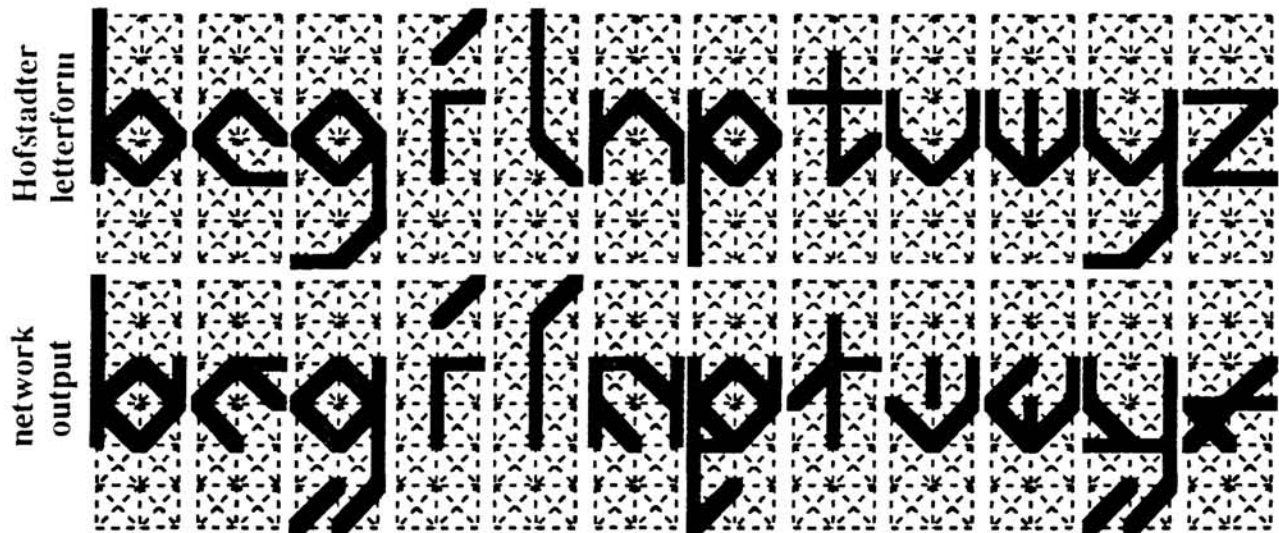

**Figure 5**: Hofstadter's letterforms from **Hunt four** font (above), and the output of our network (below) for the twelve letterforms that had never been presented during training. Hofstadter's letterforms serve merely as a guide; it is not necessary that the network reproduce these exactly to be judged successful.

Analysis of learned connection strengths (Grebert et al., 1992) reveals that different internal representations were formed for letter and for font characteristics, and that these are appropriate to the task at hand. The particular letter hidden unit shown in Figure 6 effectively "shuts down" any activity in the ascender region. Such a hidden unit would be useful when

generating **a, c, e**, etc.  Indeed this hidden unit receives strong input from all letters that have no ascenders.  The particular font hidden unit shown in Figure 6 leads to excitation of the "loop" in **Slant** font, and is used in the generation of **o, b, d, g**, etc. in that font.  We note further that our network integrated style information (e.g., the diamond shape of the "loop" for the **b, g**, the "dot" for the **l**, etc.) with the form information appropriate to the particular letter being generated.

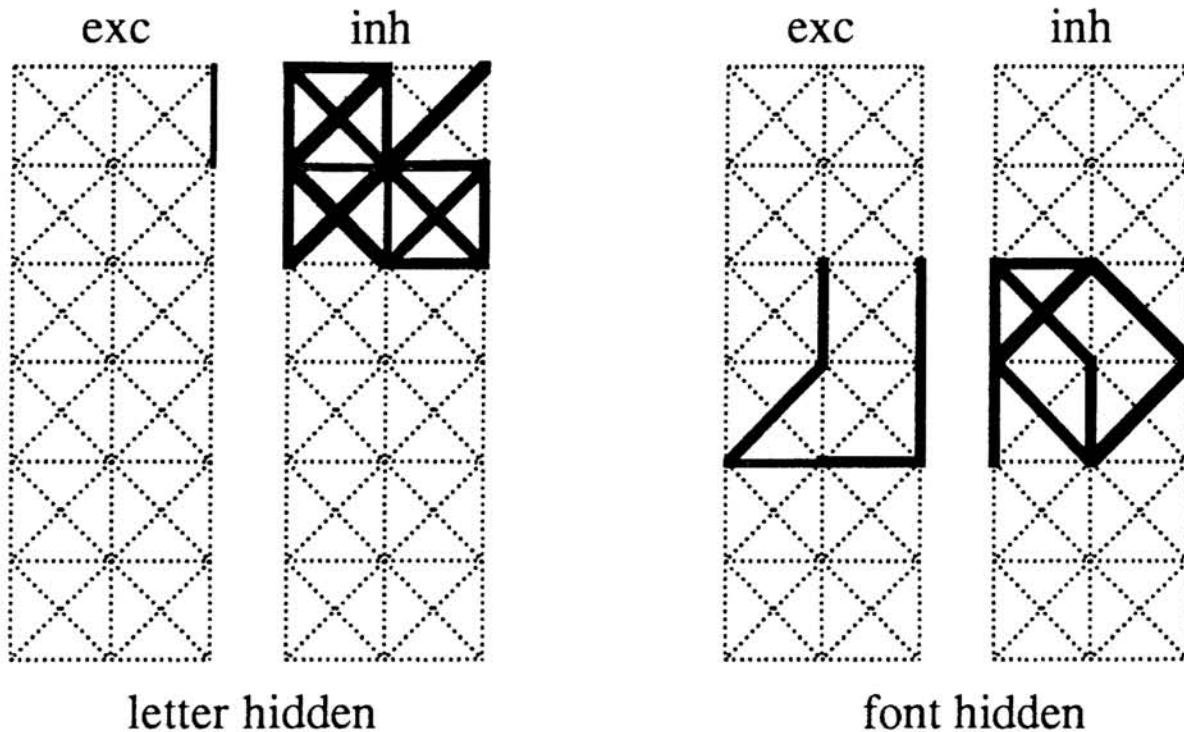

Figure 6:  Hidden unit representation for a single letter hidden unit (left) and font hidden unit (right).

In general, the network does quite well.  The only letterform quite poorly represented is **z**.  Evidently, the **z** letterform cannot be inferred from other information, presumably because **z** does not consist of any of the simplest fundamental features that make up a wide variety of other letters (left or right ascenders, loops, crosses for **t** and **f**, dots, right or left descenders).

The average adult has seen perhaps as many as $10^6$ distinct examples of each letter in perhaps $10^{10}$ presentations; in contrast, our network experienced just five or six distinct examples of each letter in $10^4$ presentations.  Out of this tremendous number of letterforms, the human virtually never experiences a **g** that has a disconnected descender (to take one example), and would not have made the errors our network does.  We suspect that the errors our network makes are similar to those a typical westerner would exhibit in generating novel characters in a completely foreign alphabet, such as Thai.  Although our network similarly has experienced only **g**'s with connected descenders, it has a very small database over which to generalize; it is to be expected, then, that the network has not yet "deduced" the connectivity constraint for **g**.  Indeed, it is somewhat surprising that our network performs as well as it does, and this gives us confidence that the architecture of Figure 2 is appropriate for the production task.

This conclusion is supported by the fact that alternative architectures gave very poor results. For instance a standard three-level backpropagation network produced illegible letterforms. Likewise, if the direct connections between letter hidden units and the output units in Figure 2 were removed, generalization performance was severely compromised.

Our network parameters could have been "fine tuned" for improved performance but such fine tuning would be appropriate for our problem alone, and not the *general* class of production problems. Even without such fine tuning, though, it is clear that the architecture of Figure 2 can successfully learn invariant features of both letter and font information, and integrate them for meaningful production of unseen letterforms. We believe this architecture can be applied to related problems, such as speech production, graphic image generation, etc.

## ACKNOWLEDGEMENTS

Thanks to David Rumelhart and Douglas Hofstadter for useful discussions. Reprint requests should be addressed to Dr. Stork at the above address, or stork@crc.ricoh.com.

## REFERENCES

Grebert, Igor, David G. Stork, Ron Keesing and Steve Mims, "Connectionist generalization for production: An example from GridFont," *Neural Networks* (1992, in press).

Hofstadter, Douglas, "Analogies and Roles in Human and Machine Thinking," Chapter 24, 547-603 in *Metamagical Themas: Questing for the Essence of Mind and Pattern* Basic Books (1985).

Margolis, Howard, *Patterns, Thinking, and Cognition: A Theory of Judgment* U. Chicago Press (1987).

Rumelhart, David E., Geoffrey E. Hinton and Ron J. Williams, "Learning Internal Representations by Error Propagation," Chapter 8, pp. 318-362 in *Parallel Distributed Processing: Explorations in the Microstructure of Cognition. Vol 1: Foundations* D. E. Rumelhart, and J. L. McClelland (eds.) MIT Press (1986).

Todd, Peter M., "A Connectionist approach to algorithmic composition," *Computer Music Journal*, **13**(4), 27-43, Winter 1989.